# AN ANALOG SELF-ORGANIZING NEURAL NETWORK CHIP

James R. Mann

Sheldon Gilbert

MIT Lincoln Laboratory
244 Wood Street
Lexington, MA 02173-0073

4421 West Estes
Lincolnwood, IL 60646

## ABSTRACT

A design for a fully analog version of a self-organizing feature map neural network has been completed. Several parts of this design are in fabrication. The feature map algorithm was modified to accommodate circuit solutions to the various computations required. Performance effects were measured by simulating the design as part of a frontend for a speech recognition system. Circuits are included to implement both activation computations and weight adaption or learning. External access to the analog weight values is provided to facilitate weight initialization, testing and static storage. This fully analog implementation requires an order of magnitude less area than a comparable digital/analog hybrid version developed earlier.

## INTRODUCTION

This paper describes an analog version of a self-organizing feature map circuit. The design implements Kohonen's self-organizing feature map algorithm [Kohonen, 1988] with some modifications imposed by practical circuit limitations. The feature map algorithm automatically adapts connection weights to nodes in the network such that each node comes to represent a distinct class of features in the input space. The system also self-organizes such that neighboring nodes become responsive to similar input classes. The prototype circuit was fabricated in two parts (for testability); a 4 node, 4 input synaptic array, and a weight adaptation and refresh circuit. A functional simulator was used to measure the effects of design constraints. This simulator evolved with the design to the point that actual device characteristics and process statistics were incorporated. The feature map simulator was used as a front-end processor to a speech recognition system whose error rates were used to monitor the effects of parameter changes on performance.

This design has evolved over the past two years from earlier experiments with a perceptron classifier [Raffel, 1987] and an earlier version of a self-organizing feature map circuit [Mann, 1988]. The perceptron classifier used a connection matrix built with multiplying D/A converters to perform the product operation for the sum-of-products computation common to all neural network algorithms. The feature map circuit also used MDAC's to perform a more complicated calculation to realize a squared Euclidean distance measure. The weights were also stored digitally, but in a unary encoded format to simplify the weight adjustment operation. This circuit contained all of the control necessary to perform weight adaptation, except for selecting a maximum responder.

The new feature map circuit described in this paper replaces the digital weight storage with dynamic analog charge storage on a capacitor. This paper will describe the circuitry and discuss problems associated with this approach to neural network implementations.

## ALGORITHM DESCRIPTION

The original Kohonen algorithm is based on a network topology such as shown in Figure 1. This illustrates a linear array of nodes, consistent with the hardware implementation being described.

Each node in the circuit computes a level of activity [Dj(t)] which indicates the similarity between the current input vector [Xi(t)] and its respective weight vector [Wij(t)]. Traditionally this would be the squared Euclidean distance given by the activation equation in the figure. If the inputs are normalized, a dot product operation can be substituted. The node most representative of the current input will be the one with the minimum or maximum output activity (classification), depending on which distance measure is used. The node number of the min./max. responder [j*] then comes to represent that class of which the input is a member. If the network is still in its learning phase, an adaptation process is invoked. This process updates the weights of all the nodes lying within a prescribed neighborhood [NEjj*(t)] of the selected node. The weights are adjusted such that the distance between the input and weight vector is diminished. This is accomplished by decreasing the individual differences between each component pair of the two vectors. The rate of learning is controlled by the gain term [a(t)]. Both the neighborhood and gain terms decrease during the learning process, stopping when the gain term reaches 0.

The following strategy was selected for the circuit implementation. First, it was assumed that inputs are normalized, thereby permitting the simpler dot product operation to be adopted. Second, weight adjustments were reduced to a simple increment/decrement operation determined by the sign of the difference between the components of the input and weight vector. Both of these simplifications were tested in the simulations described earlier and had negligible effects on overall performance as a speech vector quantizer. In addition, the prototype circuits of the analog weight version of the feature map vector quantizer do not include either the max. picker or the neighborhood operator. To date, a version of a max. picker has not yet been chosen, though many forms exist. The neighborhood operator was included in the previous version of this design, but was not repeated on this first pass.

## HARDWARE DESCRIPTION

## SYNAPTIC ARRAY

A transistor constitutes the basic synaptic connection used in this design. An analog input is represented by a voltage v(Xi) on the drain of the transistor. The weight is stored as charge q(Wij) on the gate of the transistor. If the gate voltage exceeds the maximum input voltage by an amount greater than the transistor threshold voltage, the device will be operating in the ohmic region. In this region the current [i(Dj)] through the transistor is proportional to the product of the input and weight voltages. This effectively computes one contribution to the dot product. By connecting many synapses to a single wire, current summing is performed, in accordance with Kirchoff's current law, producing the desired sum of products activity.

Figure 2 shows the transistor current as a function of the input and weight voltages. These curves merely serve to demonstrate how a transistor operating in the ohmic region will approximate a product operation.

As the input voltage begins to approach the saturation region of the transistor, the curves begin to bend over. For use in competitive learning networks, like the feature map algorithm, it is only important that the computation be monotonically increasing. These curves were the characteristics of the computation used in the simulations. The absolute values given for output current do not reflect those produced in the actual circuit.

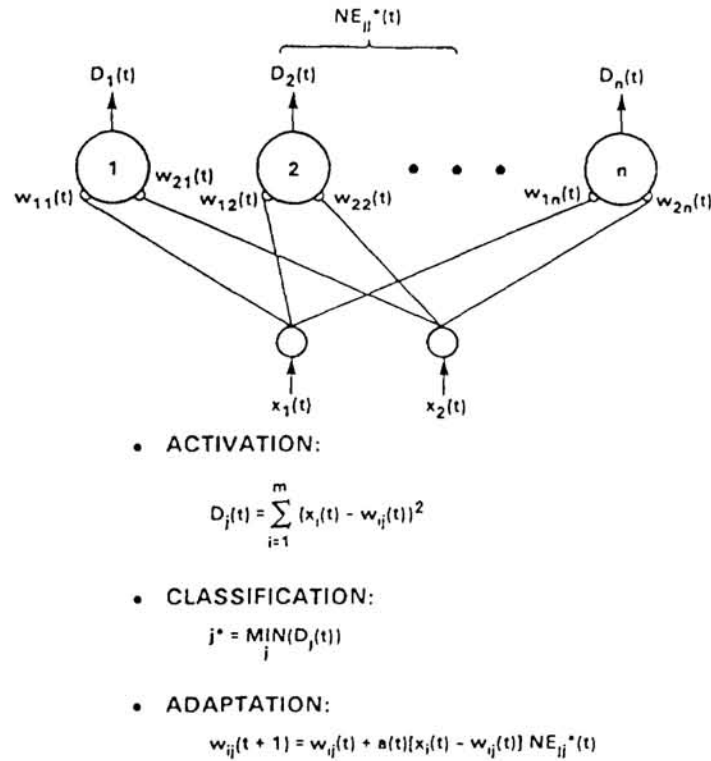

- ACTIVATION:

$$D_j(t) = \sum_{i=1}^{m} (x_i(t) - w_{ij}(t))^2$$

- CLASSIFICATION:

$$j^* = \underset{j}{MIN}(D_j(t))$$

- ADAPTATION:

$$w_{ij}(t + 1) = w_{ij}(t) + a(t)[x_i(t) - w_{ij}(t)]\, NE_{ji}^*(t)$$

Figure 1. Description of Kohonen's original feature map algorithm using a linear array of nodes.

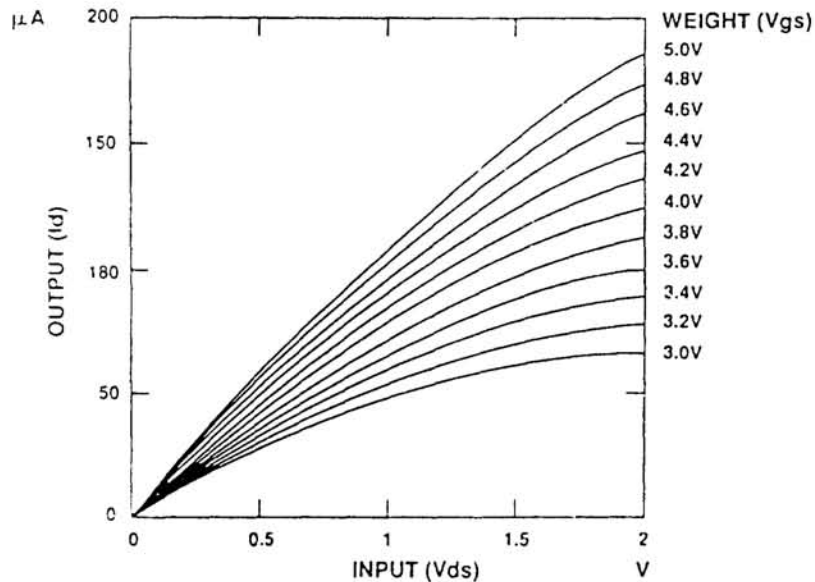

Figure 2. Typical I-V curves for a transistor operating in the ohmic region.

It should also be noted that there is no true zero weight; even the zero weight voltage contributes to the output current. But again, in a competitive network, it is only important that it contribute less than a higher weight value at that same input voltage.

In short, neither small non-linearities nor offsets interfere with circuit operation if the synapse characteristic is monotonic with weight value and input.

## SYSTEM

Figure 3 is a block diagram of the small four-node hardware prototype. The nodes are oriented horizontally, their outputs identified as I0 through I3 along the right-hand edge, representing the accumulated currents. The analog inputs [X3-X0] come in from the bottom and, traveling vertically, make connections with each node at the boxes identified as synapses. Each synapse performs its product operation between the analog weight stored at that node and the input potential.

Along the top and left sides are the control circuits for accessing weight information. The two storage registers associated with each synapse are the control signals used to select the reading and writing of weights. Weights are accessed serially by connecting to a global read and write wire, W- and W+ respectively. Besides the need for modification, the weights also drift with time, much like DRAM storage, and therefore must be refreshed periodically. This is also performed by the adaptation circuit that will be presented separately.

Control is provided by having a single "1" bit circulating through the DRAM storage bits associated with each synapse. This process goes on continuously in the background after being initialized, in parallel with the activity calculations. If the circuit is not being trained, the adaptation circuit continues to refresh the existing weights.

## WEIGHT MODIFICATION & REFRESH

A complete synapse, along with the current to voltage conversion circuit used to read the weight contents, is shown in Figure 4. The current synapse is approximately the size of two 6 transistor static RAM bits. This approximation will be used to make synaptic population estimates from current SRAM design experience. The six transistors along the top of the synapse circuit are two, three-transistor dynamic RAM cells used to control access to weight contents. These are represented in Figure 3 as the two storage elements associated with each synapse and are used as described earlier.

## READING THE WEIGHT

The two serial, vertically oriented transistors in the synapse circuit are used to sense the stored weight value. The bottom (sensing) transistor's channel is modulated by the charge stored on the weight capacitor. The sensing transistor is selected through the binary state of the 3T DRAM bit immediately above it. These two transistors used for reading the weight are duplicated in the output circuit shown to the right of the synapse. The current produced in the global read wire through the sensing transistor, is set up in the cascode current mirror arrangement in the output circuit. A mirrored version of the current, leaving the right hand side of the cascode mirror, is established in the duplicate transistor pair. The gate of this transistor is controlled by the operational amplifier as shown, and must be equivalent to the weight valueat the connection being read, if the drains are both at the same potential. This is guaranteed by the cascode mirror arrangement selected, and is set by the minus input to the amplifier.

## WRITING THE WEIGHT

The lone horizontal transistor at the bottom right corner of the synapse circuit is the weight access transistor. This connects the global write wire[W+] to the weight capacitor [Wij]. This

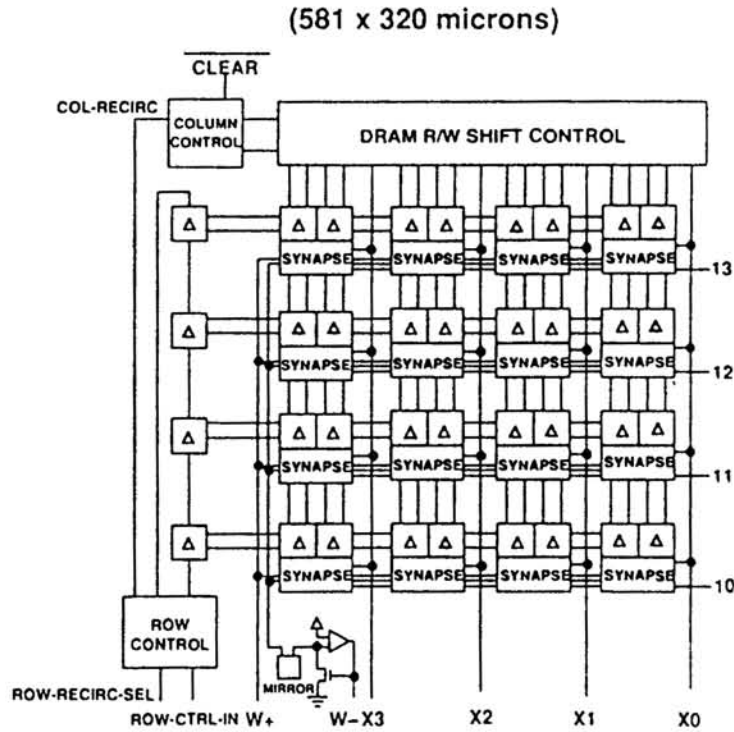

Figure 3.  A block diagram of the 4 x 4 synaptic array integrated circuit.

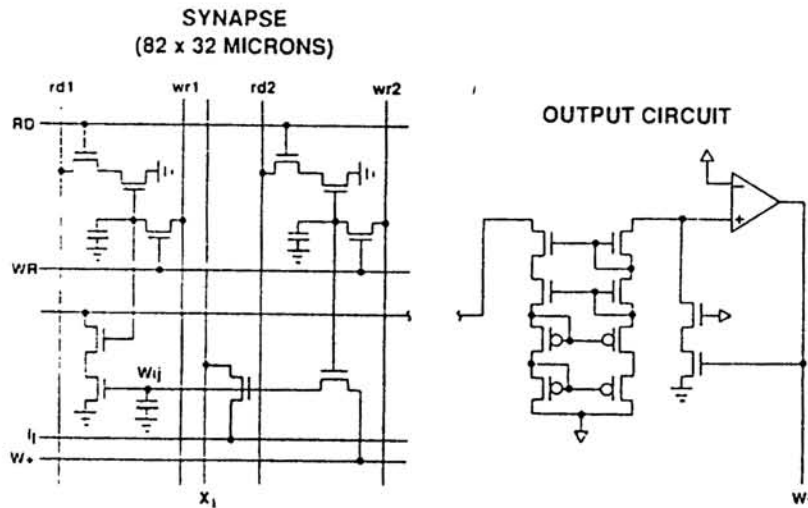

Figure 4.  Full synapse circuit.  Activation transistor is at bottom central position in the synapse circuit.

occurs whenever the DRAM element directly above it is holding a "1". When the access transistor is off, current leakage takes place, causing the voltage on the capacitor to drift with time.

There are two requirements on the weight drift for our application; That drift rates be as slow as possible, and that they drift in a known direction, in our case, toward ground. This is true because the refresh mechanism always raises the voltage to the top of a quantized voltage bin.

A cross-section of the access transistor in Figure 5 identifies the two major leakage components; reverse diode leakage to the grounded substrate (or p-well) [Io], and subthreshold channel conduction to the global write wire[Id]. The reverse diode leakage current is proportional to the area of the diffusion while the channel conduction leakage is proportional to the channel W/L ratio. Maintaining a negative voltage drift can be accomplished by sizing the devices such that reverse diode leakage dominates the channel conduction. This however would degrade the overall storage performance, and hence the minimum refresh cycle time. This can be relaxed by the technique of holding the global write line at some low voltage during everything but write cycles. This then makes the average voltage seen across the channel less than the minimum weight voltage, always resulting in a net voltage drop.

Also, these leakage currents are exponentially dependent on temperature and can be decreased by an order of magnitude with just 10's of degrees of cooling [Schwartz, 1988].

## WEIGHT REPRESENTATION

Weights, while analog, are restricted to discrete voltages. This permits the stored voltage to drift by a restricted amount (a bin), and still be refreshed to its original value. The drift rate just discussed, combined with the bin size (determined by the levels of quantization (i.e. 'of bins) and weight range (i.e. column height)), determines the refresh cycle time. The refresh cycle time, in turn, determines how many synapses (or weights) can be served by a single adaptation circuit. This means that doubling the range of the weight voltage would permit either doubling the number of quantization levels or doubling the number of synapses served by one adaptation circuit.

Weight adjustments during learning involve raising or lowering the current weight voltage to the bins immediately above or below the current bin. This constitutes a digital increment or decrement operation.

## ADAPTATION CIRCUITRY

Weight adjustments are made based upon a comparison between the current weight value and the input voltage connected to that weight. But, as these two ranges are not coincident, the comparison is made between two binary values produced by parallel flash A/D converters [Brown, 1987]. The two opposing A/D converters in Figure 6, produce a 1-of-N code, used in the comparison. The converters are composed of two stages to conserve area. The first stage performs a coarse conversion which in turn drives the upper and lower rails of the second stage converter. The selection logic decides which of the voltages among those in the second stage weight conversion circuit to route back on the global write wire [W+].
This configuration provides an easy mechanism for setting the ranges on both the inputs and weights. This is accomplished merely by setting the desired maximum and minimum voltages desired on the respective conversion circuits ([Xmin,Xmax] [Wmin,Wmax]).

## TEST RESULTS

Both circuits were fabricated in MOSIS. The synaptic array was fabricated in a 3 micron 2 metal CMOS process while the adaptation circuitry was fabricated in a similar 2 micron process. To date, only the synaptic array has been tested. In these tests, the input was restricted to a 0 to1

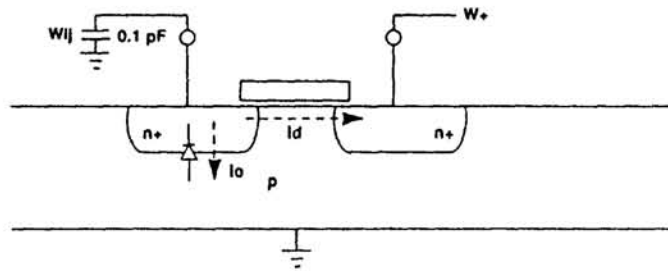

Figure 5.  Cross-sectional view of a weight access transistor with leakage
currents.

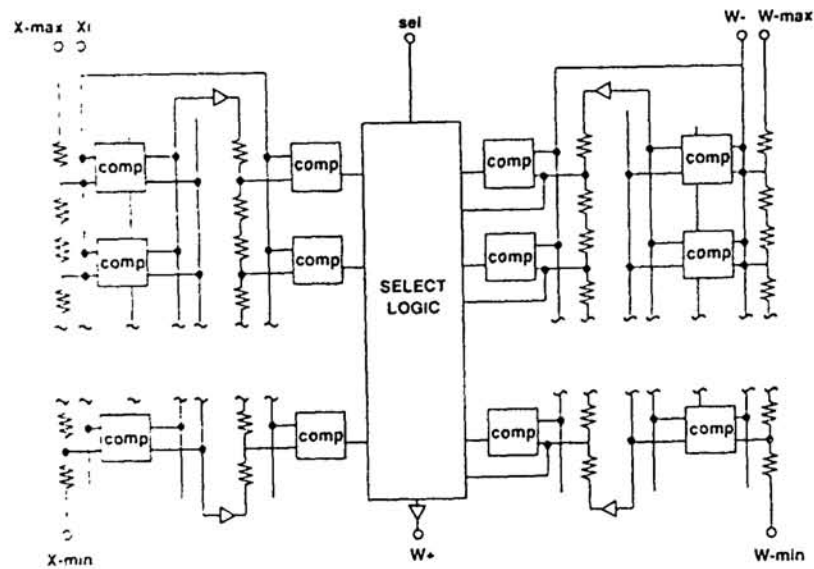

Figure 6.  Block diagram of the weight adaptation and refresh circuit.
Comparison of digital A\D outputs and new weight selection takes
place in the box marked SELECT LOGIC.

V range while the weight range was 2 to 3 V. Most of these early tests were done with binary weights, either 2 V or 3 V, corresponding to a "0" and a "1".

The synapses and associated control circuitry all work as expected. The circuit can be clocked up to 7 MHz. The curves shown in Figure 7 display a typical neuron output during two modes of operation; a set of four binary weights with all of the inputs swept together over their operating range, and a single, constant input with its weight being swept through its operating range.

The graphs in Figure 8 show the temporal behavior of the weight voltage stored at a single synapse. On the left is plotted the output current to weight voltage, for converting between the two quantities. The right hand plot is the output current of the synapse plotted against time. If the weight voltage bin size is set to 15 mV (2V range, 128 bins), a 3 to 4 second refresh cycle time limit would be required. This is a very lenient constraint and may permit a much finer quantization than expected.

The circuitry for reading the weights was tested and appears to be inoperative. The cascode mirror requires a very high potential at the p-channel sources which causes the circuit to latch up when the clocks are turned on. This circuit will be isolated and tested under static conditions.

## CONCLUSIONS

In summary, a design for an analog version of a self- organizing feature map has been completed and prototype versions of the synaptic array and the adaptation circuitry have been fabricated. The devices are still undergoing testing and characterization, but the basic DRAM control and synaptic operation have been demonstrated. Simulations have provided the guidance on design choices. These have been instrumental in providing information on effects due to quantization, computational non-linearities, and process variations. The new design offers a significant increase in density over a digital/analog hybrid approach. The 84 pin standard frame package from MOSIS will accommodate more than 8000 synapses of from 6 to 8 bits accuracy. It appears that control modifications may offer even greater densities in future versions.

This work was sponsored by the Department of the Air Force, and the Defense Advanced Research Projects Agency, the views expressed are those of the author and do not reflect the official policy or position of the U.S. Government.

## REFERENCES

P. Brown, R. Millecchia M. Stinely. Analog Memory for Continuous-Voltage, Discrete-Time Implementation of Neural Networks. Proc. IEEE Intl. Conf. on Neural Networks. 1987.

T. Kohonen. Self-Organization and Associative Memory. Springer-Verlag. 1988.

J. Mann, R. Lippmann, R. Berger J. Raffel. A Self-Organizing Neural Net Chip. IEEE 1988 Custom Integrated Circuits Conference. pp. 10.3.1-10.3.5. 1988.

J. Raffel, J. Mann, R. Berger, A. Soares S. Gilbert. A Generic Architecture for Wafer-Scale Neuromorphic Systems. Proc. IEEE Intl. Conf. on Neural Networks.1987.

D.B. Schwartz R.E. Howard. A Programmable Analog Neural Network Chip. IEEE 1988 Custom Integrated Circuits Conference. pp. 10.2.1-10.2.4. 1988.

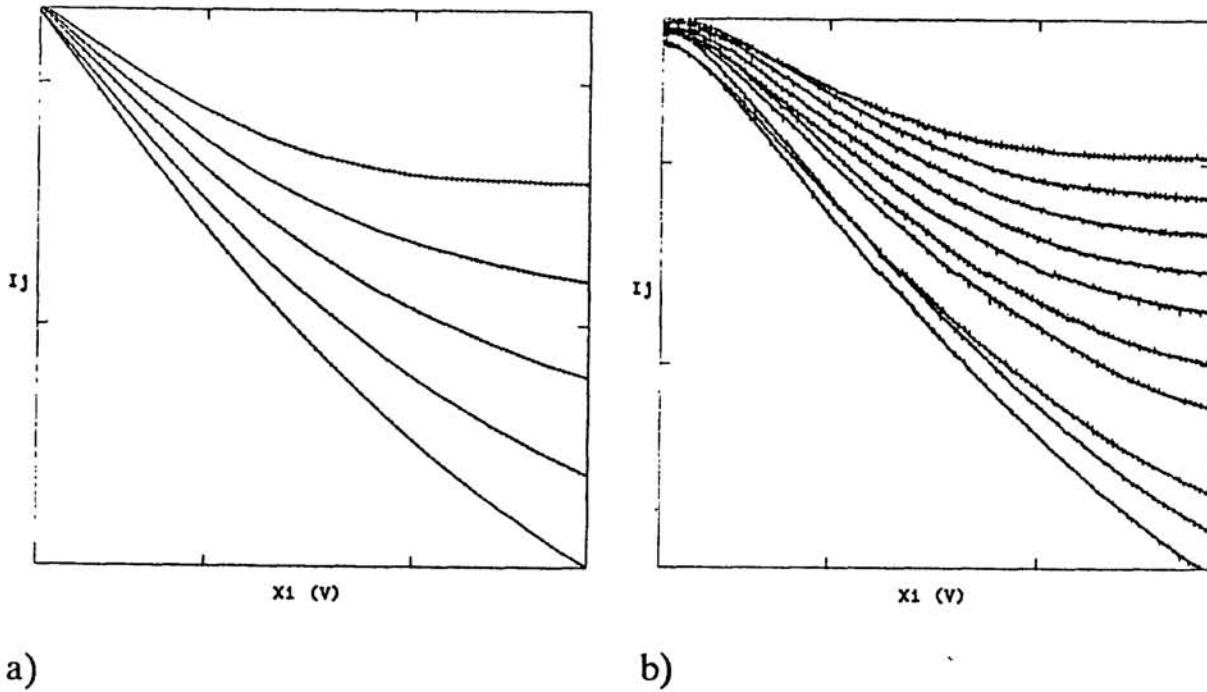

a)                                              b)

Figure 7.  a) plot of output current (Ij) as a function of imput voltage (Xi)
          between 0 and 1 volt for 0 (top curve) to 4 (bottom curve) weights
          "DN".  b) plot of output current (Ij) vs. input voltage (Xi) from
          0 to 1V for a weight voltage between 2V (top) and 3V (bottom)
          in 0.1V steps.

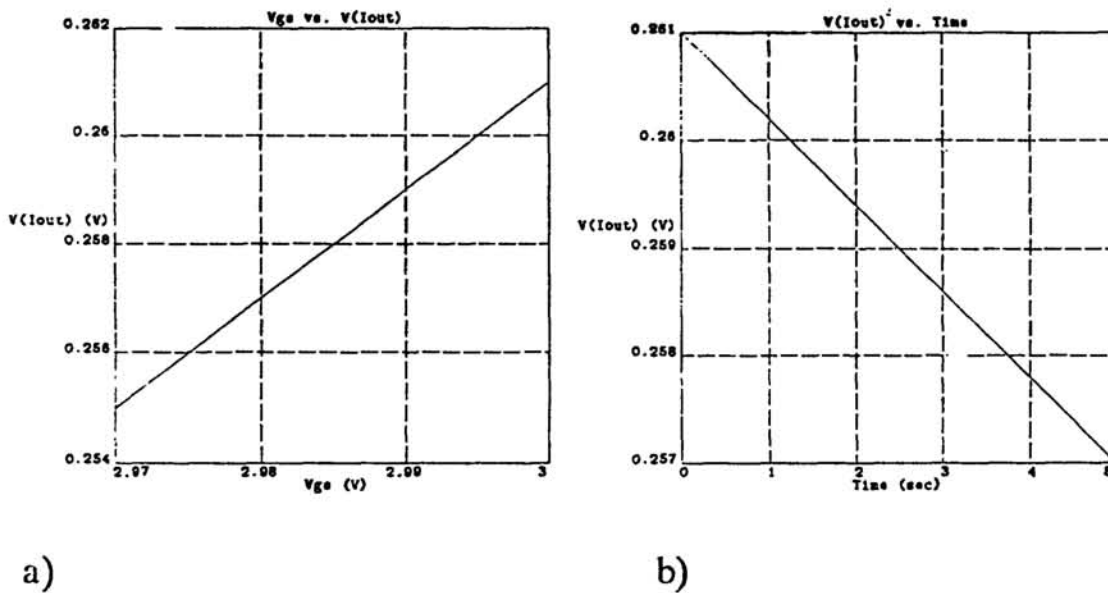

a)                                              b)

Figure 8.  a) plot of output current V (Iout) vs. weight voltage.  b) plot of
          output current as a function of time with W+ held at 0VX and the
          local weight initially set to 3V.